# NEUROMORPHIC NETWORKS BASED ON SPARSE OPTICAL ORTHOGONAL CODES

Mario P. Vecchi and Jawad A. Salehi

Bell Communications Research

435 South Street

Morristown, NJ 07960-1961

## Abstract

A family of neuromorphic networks specifically designed for communications and optical signal processing applications is presented. The information is encoded utilizing sparse Optical Orthogonal Code sequences on the basis of unipolar, binary $(0,1)$ signals. The generalized synaptic connectivity matrix is also unipolar, and clipped to binary $(0,1)$ values. In addition to high-capacity associative memory, the resulting neural networks can be used to implement general functions, such as code filtering, code mapping, code joining, code shifting and code projecting.

## 1  Introduction

Synthetic neural nets[1,2] represent an active and growing research field. Fundamental issues, as well as practical implementations with electronic and optical devices are being studied. In addition, several learning algorithms have been studied, for example stochastically adaptive systems[3] based on many-body physics optimization concepts[4,5].

Signal processing in the optical domain has also been an active field of research. A wide variety of non-linear all-optical devices are being studied, directed towards applications both in optical computating and in optical switching. In particular, the development of Optical Orthogonal Codes (OOC)[6] is specifically interesting to optical communications applications, as it has been demonstrated in the context of Code Division Multiple Access (CDMA)[7].

In this paper we present a new class of neuromorphic networks, specifically designed for optical signal processing and communications, that encode the information in *sparse* OOC's. In Section 2 we review some basic concepts. The new neuromorphic networks are defined in Section 3, and their associative memory properties are presented in Section 4. In Section 5 other general network functions are discussed. Concluding remarks are given in Section 6.

## 2  Neural Networks and Optical Orthogonal Codes

### 2.1  Neural Network Model

Neural network are generally based on multiply-threshold-feedback cycles. In the Hopfield model[2], for instance, a connectivity $\overline{T}$ matrix stores the M different memory elements, labeled $m$, by the sum of outer products,

$$T_{ij} = \sum_{m}^{M} u_i^m u_j^m \; ; \;\; i,j = 1,2...N \tag{1}$$

where the state vectors $\underline{u}^m$ represent the memory elements in the bipolar $(-1, 1)$ basis. The diagonal matrix elements in the Hopfield model are set to zero, $T_{ii} = 0$.

For a typical memory recall cycle, an input vector $\underline{v}^{in}$, which is close to a particular memory element $m = k$, multiplies the $\overline{T}$ matrix, such that the output vector $\underline{v}^{out}$ is given by

$$\hat{v}_i^{out} = \sum_{j=1}^{N} T_{ij} v_j^{in} \; ; \; i, j = 1, 2 ... N \tag{2}$$

and can be seen to reduce to

$$\hat{v}_i^{out} \approx (N-1) u_i^k + \sqrt{(N-1)(M-1)} \tag{3}$$

for large N and in the case of randomly coded memory elements $\underline{u}^m$.

In the Hopfield model, each output $\hat{v}^{out}$ is passed through a thresholding stage around zero. The thresholded output signals are then fed back, and the multiply and threshold cycle is repeated until a final stable output $\underline{v}^{out}$ is obtained. If the input $\underline{v}^{in}$ is sufficiently close to $\underline{u}^k$, and the number of state vectors is small (i.e. $M \ll N$), the final output will converge to memory element $m = k$, that is, $\underline{v}^{out} \rightarrow \underline{u}^k$. The associative memory property of the network is thus established.

## 2.2 Optical Orthogonal Codes

The OOC sequences have been developed[6,7] for optical CDMA systems. Their properties have been specifically designed for this purpose, based on the following two conditions: each sequence can be easily distinguished from a shifted version of itself, and each sequence can be easily distinguished from any other shifted or unshifted sequence in the set. Mathematically, the above two conditions are expressed in terms of auto- and crosscorrelation functions. Because of the non-negative nature of optical signals[1], OOC are based on unipolar $(0, 1)$ signals[7].

In general, a family of OOC is defined by the following parameters:

- F, the *length* of the code,

- K, the *weight* of the code, that is, the number of 1's in the sequence,

- $\lambda_a$, the auto-correlation value for all possible shifts, other than the zero shift,

- $\lambda_c$, the cross-correlation value for all possible shifts, including the zero shift.

For a given code length F, the maximum number of distinct sequences in a family of OOC depends on the chosen parameters, that is, the weight of the code K and the allowed overlap $\lambda_a$ and $\lambda_c$. In this paper we will consider OOC belonging to the *minimum overlap* class, $\lambda_a = \lambda_c = 1$.

# 3  Neuromorphic Optical Networks

Our neuromorphic networks are designed to take full advantage of the properties of the OOC. The connectivity matrix $\overline{T}$ is defined as a sum of outer products, by analogy with (1), but with the following important modifications:

1. The memory vectors are defined by the sequences of a given family of OOC, with a basis given by the unipolar, binary pair $(0,1)$. The dimension of the sparse vectors is given by the length of the code F, and the maximum number of available items depends on the chosen family of OOC.

2. All of the matrix elements $T_{ij}$ are clipped to unipolar, binary $(0,1)$ values, resulting in a sparse and simplified connectivity matrix, without any loss in the functional properties defined by our neuromorphic networks.

3. The diagonal matrix elements $T_{ii}$ are *not* set to zero, as they reflect important information implicit in the OOC sequences.

4. The threshold value is *not* zero, but it is chosen to be equal to K, the weight of the OOC.

5. The connectivity matrix $\overline{T}$ is generalized to allow for the possibility of a variety of outer product options: self-outer products, as in (1), for associative memory, but also cross-outer products of different forms to implement various other system functions.

A simplified schematic diagram of a possible optical neuromorphic processor is shown in Figure 1. This implementation is equivalent to an incoherent optical matrix-vector multiplier[8], with the addition of nonlinear functions. The input vector is clipped using an optical hard-limiter with a threshold setting at 1, and then it is anamorphically imaged onto the connectivity mask for $\overline{T}$. In this way, the $i^{th}$ pixel of the input vector is imaged onto the $i^{th}$ column of the $\overline{T}$ mask. The light passing through the mask is then anamorphically imaged onto a line of optical threshold elements with a threshold setting equal to K, such that the $j^{th}$ row is imaged onto the $j^{th}$ threshold element.

# 4  Associative Memory

The associative memory function is defined by a connectivity matrix $\overline{T}^{MEM}$ given by:

$$T_{ij}^{MEM} = \mathcal{G}\left\{\sum_{m}^{M} x_i^m x_j^m\right\} \quad ; \quad i,j = 1,2...F \tag{4}$$

where each memory element $\underline{x}^m$ corresponds to a given sequence of the OOC family, with code length F. The matrix elements of $\overline{T}^{MEM}$ are all clipped, unipolar values, as indicated by the function $\mathcal{G}\{\}$, such that,

$$\mathcal{G}\{\zeta\} = \begin{cases} 1 & if\ \zeta \geq 1 \\ 0 & if\ \zeta < 1 \end{cases} \tag{5}$$

We will now show that an input vector $\underline{z}^k$, which corresponds to memory element $m = k$, will produce a stable output (equal to the wanted memory vector) in a *single pass* of the multiply and threshold process.

The *multiplication* can be written as:

$$\hat{v}_i^{out} = \sum_j^F T_{ij}^{MEM} x_j^k \qquad (6)$$

We remember that the non-linear clipping function $\mathcal{G}\{\}$ is to be applied *first* to obtain $\overline{T}^{MEM}$. Hence,

$$\hat{v}_i^{out} = \sum_j^F x_j^k \; \mathcal{G}\left\{ x_i^k x_j^k + \sum_{m \neq k}^M x_i^m x_j^m \right\} \qquad (7)$$

For $x_i^k = 0$, only the second term in (7) contributes, and the pseudo-orthogonality properties of the OOC allow us to write:

$$\sum_{j \neq i}^F x_j^k \; \mathcal{G}\left\{ \sum_{m \neq k}^M x_i^m x_j^m \right\} \leq \lambda_c, \qquad (8)$$

where the cross-correlation value is $\lambda_c < K$.

For $x_i^k = 1$, we again consider the properties of the OOC to obtain for the first term of (7):

$$\sum_j^F x_j^k x_i^k x_j^k = K x_i^k, \qquad (9)$$

where K is the weight of the OOC.

Therefore, the result of the multiplication operation given by (7) can be written as:

$$\hat{v}_i^{out} = K x_i^k + \begin{bmatrix} \textit{value strictly} \\ \textit{less than K} \end{bmatrix} \qquad (10)$$

The *thresholding* operation follows, around the value K as explained in Section 3. That is, (10) is thresholded such that:

$$v_i^{out} = \begin{cases} 1 & \text{if } \hat{v}_i^{out} \geq K \\ 0 & \text{if } \hat{v}_i^{out} < K, \end{cases} \qquad (11)$$

hence, the final output at the end of a *single pass* will be given by: $v_i^{out} = x_i^k$.

The result just obtained can be extended to demonstrate the single pass convergence when the input vector is close, but not necessarily equal, to a stored memory element. We can draw the following conclusions regarding the properties of our neuromorphic networks based on OOC:

- For any given input vector $\underline{v}^{in}$, the single pass output will correspond to the memory vector $\underline{z}^m$ which has the smallest Hamming distance to the input.

- If the input vector $\underline{v}^{in}$ is missing a single 1-element from the K 1's of an OOC, the single pass output will be the null or zero vector.

- If the input vector $\underline{v}^{in}$ has the same Hamming distance to two (or more) memory vectors $\underline{z}^m$, the single pass output will be the logical sum of those memory vectors.

The ideas just discussed were tested with a computer simulation. An example of associative memory is shown in Table 1, corresponding to the OOC class of length $F = 21$ and weight $K = 2$. For this case, the maximum number of independent sequences is $M = 10$. The connectivity matrix $\overline{T}^{MEM}$ is seen in Table 1, where one can clearly appreciate the simplifying features of our model, both in terms of the sparsity and of the unipolar, clipped values of the matrix elements. The computer simulations for this example are shown in Table 2. The input vectors $\underline{a}$ and $\underline{b}$ show the error-correcting memory recovery properties. The input vector $\underline{c}$ is equally distant to memory vectors $\underline{z}^3$ and $\underline{z}^8$, resulting in an output which is the sum $(\underline{z}^3 \oplus \underline{z}^8)$. And finally, input vector $\underline{d}$ is closest to $\underline{z}^1$, but one 1 is missing, and the output is the zero vector. The mask in Figure 1 shows the optical realization of the Table 1, where the transparent pixels correspond to the 1's and the opaque pixels to the 0's of the connectivity matrix $\overline{T}^{MEM}$.

It should be pointed out that the capacity of our network is significant. From the previous example, the capacity is seen to be $\approx F/2$ for *single pass* memory recovery. This result compares favorably with the capacity of a Hopfield model[9], of $\approx F/4 \ln F$.

## 5 General Network Functions

Our neuromorphic networks, based on OOC, can be generalized to perform functions other than associative memory storage by constructing non-symmetrical connectivity matrices. The single pass convergence of our networks avoids the possibility of limit-cycle oscillations. We can write in general:

$$T_{ij} = \mathcal{G}\left\{ \sum_{m=1}^{M} y_i^m x_j^m \right\}, \tag{12}$$

where each pair defined by m includes two vectors $\underline{y}^m$ and $\underline{z}^m$, which are *not necessarily equal*. The clipping function $\mathcal{G}\{\}$ insures that all matrix elements are binary (0,1) values. The possible choice of vector pairs is not completely arbitrary, but there is a wide variety of functions that can be implemented for each family of OOC. We will now discuss some of the applications that are of particular interest in optical communication systems.

### 5.1 Code Filtering (CDMA)

Figure 2 shows an optical CDMA network in a star configuration. M nodes are interconnected with optical fibers to a passive MxM star coupler that broadcasts the optical signals. At each node there is a data encoder that maps each bit of information to the OOC sequence corresponding to the user for which the transmission is intended. In addition, each node has a filter and decoder that recognizes its specific OOC sequence. The optical transmission rate has been expanded by a factor F corresponding to the length of the OOC sequence. Within the context of a CDMA communication system[7], the filter or decoder must perform the function of recognizing a specific OOC sequence in the presence of other interfering codes sent on the common transmission medium.

We can think, then, of one of our neuromorphic networks as a filter, placed at a given receiver node, that will recognize the specific code that it was programmed for.

We define for this purpose a connectivity matrix as

$$T_{ij}^{CDMA} = x_i^k x_j^k \ ; \ i,j = 1,2...F,$$ (13)

where only one vector $\underline{x}^k$ is stored at each node. This symmetric, clipped connectivity matrix will give an output equal to $\underline{x}^k$ whenever the input contains this vector, and a null or zero output vector otherwise. It is clear by comparing (13) with (4) that the CDMA filtering matrix is equivalent to an associative memory matrix with only one item imprinted in the memory. Hence the discussion of Section 4 directly applies to the understanding of the behaviour of $\overline{T}^{CDMA}$.

In order to evaluate the performance of our neuromorphic network as a CDMA filter, computer simulations were performed. Table 3 presents the $\overline{T}^{CDMA}$ matrix for a particular node defined by $\underline{x}^k$ of a CDMA system based on the OOC family $F = 21$, $K = 2$. The total number of distinct codes for this OOC family is $M = 10$, hence there are 9 additional OOC sequences that interfere with $\underline{x}^k$, labeled in Table 3 $\underline{x}^1$ to $\underline{x}^9$.

The performance was simulated by generating random composite sequences from the set of codes $\underline{x}^1$ to $\underline{x}^9$ arbitrarily shifted. All inputs are unipolar and clipped (0,1) signals. The results presented in Table 4 give examples of our simulation for the $\overline{T}^{CDMA}$ matrix shown in Table 3. The input $\underline{a}$ is the (logical) sum of a 1-bit (vector $\underline{x}^k$), plus interfering signals from arbitrarily shifted sequences of $\underline{x}^2$, $\underline{x}^3$, $\underline{x}^4$, $\underline{x}^6$ and $\underline{x}^9$. The output of the neuromorphic network is seen to recover accurately the desired vector $\underline{x}^k$. The input vector $\underline{b}$ contains a 0-bit (null vector), plus the shifted sequences of $\underline{x}^1$, $\underline{x}^2$, $\underline{x}^3$, $\underline{x}^6$, $\underline{x}^7$ and $\underline{x}^8$, and we see that the output correctly recovers a 0-bit.

As discussed in Section 4, our neuromorphic network will always correctly recognize a 1-bit (vector $\underline{x}^k$) presented to its input. On the other hand [2], there is the possibility of making an error when a 0-bit is sent, and the interfering signals from other nodes happen to generate the chip positions of $\underline{x}^k$. This case is shown by input vector $\underline{c}$ of Table 4, which contains a 0-bit (null vector), plus shifted sequences of $\underline{x}^2$, $\underline{x}^3$, $\underline{x}^4$, $\underline{x}^5$, $\underline{x}^6$, $\underline{x}^7$ and $\underline{x}^8$ in such a way that the output is erroneously given as a 1-bit. The properties of the OOC sequences are specifically chosen to minimize these errors[7], and the statistical results of our simulation are also shown in Table 4. It is seen that, as expected, when a 1-bit is sent it is always correctly recognized. On the other hand, when 0-bits are sent, occasional errors occur. Our simulation, yields an overall bit error rate $(BER)$ of $BER_{sim} = 5.88\%$, as shown in Table 4.

These results can be compared with theoretical calculations[7] which yield an estimate for the $BER$ for the CDMA system described:

$$BER_{calc} \approx \frac{1}{2} \prod_{k=0}^{K-1} \left[ 1 - q^{M-1-k} \right],$$ (14)

where $q \equiv 1 - \frac{K}{2F}$. For the example of the OOC family $F = 21$, $K = 2$, with $M = 10$, the above expression yields $BER_{calc} \approx 5.74\%$.

It is seen, therefore, that our neuromorphic network approaches the minimum possible $BER$ for a given family of OOC. In fact, the results obtained using our $\overline{T}^{CDMA}$ are equivalent CDMA detection scheme based on "optical-AND-gates"[10], which corresponds to the limiting $BER$ determined by the properties of the OOC themselves[3]. The optical mask corresponding to the code filtering function is shown in Figure 3.

## 5.2 Other Functions

As a first example of a non-symmetric $\overline{T}$ matrix, let us consider the function of *mapping* an input code to a corresponding different output code. We define our mapping matrix as:

$$T_{ij}^{MAP} = \mathcal{G}\left\{\sum_m y_i^m x_j^m\right\} \quad ; \quad i,j = 1,2...F, \tag{15}$$

where an input vector $\underline{x}^m$ will produce a *different* output vector code $\underline{y}^m$.

The function of code *joining* is defined by a transfer function that takes a given input code and produces at the output a chosen combination of two or more codes. This function is performed by expressing the general matrix given by 12 as follows:

$$T_{ij}^{JOIN} = \mathcal{G}\left\{\sum_m (y_i^m + w_i^m + ...)x_j^m\right\} \quad ; \quad i,j = 1,2...F, \tag{16}$$

where an input vector $\underline{x}^m$ will result in an output that joins several vector codes ($\underline{y}^m \oplus \underline{w}^m \oplus ...$).

The code *shifting* matrix $\overline{T}^{SHIFT}$ will allow for the shift of a given code sequence, such that both input and output correspond to the same code, but shifted with respect to itself. That is,

$$T_{ij}^{SHIFT} = \mathcal{G}\left\{\sum_m x(s)_i^m x(0)_j^m\right\} \quad ; \quad i,j = 1,2...F, \tag{17}$$

where we have indicated an unshifted code sequence by $\underline{x}(0)^m$, and its corresponding output pair as a shifted version of itself $\underline{x}(s)^m$.

The code *projecting* function corresponds to processing an input vector that contains the logical sum of several codes, and projecting at the output a selected single code sequence. The corresponding matrix $\overline{T}^{PROJ}$ is given by:

$$T_{ij}^{PROJ} = \mathcal{G}\left\{\sum_m x_i^m(y_j^m + w_j^m + ...)\right\} \quad ; \quad i,j = 1,2...F, \tag{18}$$

where each input vector ($\underline{y}^m \oplus \underline{w}^m \oplus ...$) will project at the output to a single code $\underline{x}^m$. In general, the resulting output code sequence $\underline{x}^m$ could correspond to a code *not necessarily* contained in the input vector.

The performance and error correcting properties of these, and other, general functions follow a similar behaviour as discussed in Section 4.

# 6 Conclusions

The neuromorphic networks presented, based on sparse Optical Orthogonal Code (OOC) sequences, have been shown to have a number of attractive properties. The unipolar, clipped nature of the synaptic connectivity matrix simplifies the implementation. The single pass convergence further allows for general network functions that are expected to be of particular interest in communications and signal processing systems.

The coding of the information, based on OOC, has also been shown to result in high capacity associative memories. The combination of efficient associative memory properties, plus a variety of general network functions, also suggests the possible application of our neuromorphic networks in the implementation of computational functions based on optical symbolic substitution.

The family of neuromorphic networks discussed here emphasizes the importance of understanding the general properties of non-negative systems based on sparse codes[11]. It is hoped that our results will stimulate further work on the fundamental relationship between coding, or representations, and the information processing properties of neural nets.

# Acknowledgement

We thank J. Y. N. Hui and J. Alspector for many useful discussions, and C. A. Brackett for his support and encouragement of this research.

## Footnotes

[1]We refer to optical *intensity* signals, and not to detection systems sensitive to phase information.

[2]Our channel can be described, then, as a binary Z-channel between each two nodes dynamically establishing a communication path

[3]The $BER$ for the OOC family shown in this example are far too large for a useful CDMA communications system. Our choice intended to show computer simulated results within a reasonable computation time.

# References

[1] S. Grossberg. In K. Schmitt, editor, *Delay and Functional-Differential Equations and Their Applications*, page 121, Academic Press, New York, NY, 1972.

[2] J. J. Hopfield. Neural Networks and Physical Systems with Emergent Collective Computational Abilities. *Proc. Nat. Acad. Sci. USA*, 79:2254, 1982.

[3] D. H. Ackley, G. E. Hinton, and T. J. Sejnowski. A Learning Algorithm for Boltzmann Machines. *Cogn. Sci.*, 9:147, 1985.

[4] S. Kirkpatrick, C. D. Gelatt, and M. P. Vecchi. Optimization by Simulated Annealing. *Science*, 220:671, 1983.

[5] M. P. Vecchi and S. Kirkpatrick. Global Wiring by Simulated Annealing. *IEEE Trans. CAD of Integrated Circuits and Systems*, CAD-2:215, 1983.

[6] F. R. K. Chung, J. A. Salehi, and V. K. Wei. Optical Orthogonal Codes: Design, Analysis and Applications. In *IEEE International Symposium on Information Theory, Catalog No. 86CH2374-7*, 1986. Accepted for publication in IEEE Trans. on Information Theory.

[7] J. A. Salehi and C. A. Brackett. Fundamental Principles of Fiber Optics Code Division Multiple Access. In *IEEE International Conference on Communications*, 1987.

[8] N. H. Farhat, D. Psaltis, A. Prata, and E. Paek. Optical Implementation of the Hopfield Model. *Appl. Opt.*, 24:1469, 1985.

[9] R. J. McEliece, E. C. Posner, E. R. Rodemich, and S. S. Venkatesh. The Capacity of Hopfield Associative Memory. *IEEE Trans. on Information Theory*, IT-33:461, 1987.

[10] J. A. Salehi. Principles and Applications of Optical AND Gates in Fiber Optics Code Division Multiple Access Networks. *In preparation*, 1987.

[11] G. Palm. Technical comments. *Science*, 235:1226, 1987.

Table 1: Associative Memory. Example showing storage of 10 distinct code sequences corresponding to the chosen OOC family.

**OOC Family: F = 31, K = 3**

Code Vectors:

| | | | | | | | | | | | | | | | | | | | | | | | | | | | | | | | |
|---|---|---|---|---|---|---|---|---|---|---|---|---|---|---|---|---|---|---|---|---|---|---|---|---|---|---|---|---|---|---|---|
| $c^1$ | 1 | 1 | 0 | 0 | 0 | 0 | 0 | 0 | 0 | 0 | 0 | 0 | 0 | 0 | 0 | 0 | 0 | 0 | 0 | 0 | 0 | 0 | 0 | 0 | 0 | 0 | 0 | 0 | 0 | 0 | 0 |
| $c^2$ | 0 | 0 | 1 | 0 | 1 | 0 | 0 | 0 | 0 | 0 | 0 | 0 | 0 | 0 | 0 | 0 | 0 | 0 | 0 | 0 | 0 | 0 | 0 | 0 | 0 | 0 | 0 | 0 | 0 | 0 | 0 |
| $c^3$ | 0 | 0 | 0 | 1 | 0 | 0 | 1 | 0 | 0 | 0 | 0 | 0 | 0 | 0 | 0 | 0 | 0 | 0 | 0 | 0 | 0 | 0 | 0 | 0 | 0 | 0 | 0 | 0 | 0 | 0 | 0 |
| $c^4$ | 0 | 0 | 0 | 0 | 0 | 1 | 0 | 0 | 1 | 0 | 0 | 0 | 0 | 0 | 0 | 0 | 0 | 0 | 0 | 0 | 0 | 0 | 0 | 0 | 0 | 0 | 0 | 0 | 0 | 0 | 0 |
| $c^5$ | 0 | 0 | 0 | 0 | 0 | 0 | 0 | 1 | 0 | 0 | 0 | 1 | 0 | 0 | 0 | 0 | 0 | 0 | 0 | 0 | 0 | 0 | 0 | 0 | 0 | 0 | 0 | 0 | 0 | 0 | 0 |
| $c^6$ | 0 | 0 | 0 | 0 | 0 | 0 | 0 | 0 | 0 | 1 | 0 | 0 | 0 | 0 | 0 | 0 | 0 | 1 | 0 | 0 | 0 | 0 | 0 | 0 | 0 | 0 | 0 | 0 | 0 | 0 | 0 |
| $c^7$ | 0 | 0 | 0 | 0 | 0 | 0 | 0 | 0 | 0 | 0 | 1 | 0 | 0 | 0 | 0 | 0 | 0 | 0 | 0 | 0 | 1 | 0 | 0 | 0 | 0 | 0 | 0 | 0 | 0 | 0 | 0 |
| $c^8$ | 0 | 0 | 0 | 0 | 0 | 0 | 0 | 0 | 0 | 0 | 0 | 0 | 1 | 0 | 0 | 0 | 0 | 0 | 0 | 0 | 0 | 0 | 1 | 0 | 0 | 0 | 0 | 0 | 1 | 0 | 0 |
| $c^9$ | 0 | 0 | 0 | 0 | 1 | 0 | 0 | 0 | 0 | 0 | 0 | 0 | 0 | 0 | 0 | 0 | 0 | 1 | 0 | 0 | 0 | 0 | 0 | 0 | 0 | 0 | 0 | 0 | 0 | 0 | 0 |
| $c^{10}$ | 0 | 0 | 0 | 0 | 1 | 0 | 0 | 0 | 0 | 0 | 0 | 0 | 0 | 0 | 0 | 0 | 1 | 0 | 0 | 0 | 0 | 0 | 0 | 0 | 0 | 0 | 0 | 0 | 0 | 0 | 0 |

Connectivity Matrix $T^{MEM}$:

Table 3: Code Filtering (CDMA).

**OOC Family: F = 31, K = 3**

Imprinted Code for Node k:

| | | | | | | | | | | | | | | | | | | | | | | | | | | | | | | | | |
|---|---|---|---|---|---|---|---|---|---|---|---|---|---|---|---|---|---|---|---|---|---|---|---|---|---|---|---|---|---|---|---|
| $c^1$ | 0 | 0 | 0 | 0 | 0 | 0 | 0 | 0 | 0 | 1 | 0 | 0 | 0 | 0 | 0 | 1 | 0 | 0 | 0 | 0 | 0 | 0 | 0 | 0 | 0 | 0 | 0 | 0 | 0 | 0 | 0 |

Connectivity Matrix $T^{CDMA}$:

Remaining 9 Interfering Codes:

| | | | | | | | | | | | | | | | | | | | | | | | | | | | | | | | | |
|---|---|---|---|---|---|---|---|---|---|---|---|---|---|---|---|---|---|---|---|---|---|---|---|---|---|---|---|---|---|---|---|
| $c^1$ | 1 | 1 | 0 | 0 | 0 | 0 | 0 | 0 | 0 | 0 | 0 | 0 | 0 | 0 | 0 | 0 | 0 | 0 | 0 | 0 | 0 | 0 | 0 | 0 | 0 | 0 | 0 | 0 | 0 | 0 | 0 |
| $c^2$ | 0 | 0 | 1 | 0 | 1 | 0 | 0 | 0 | 0 | 0 | 0 | 0 | 0 | 0 | 0 | 0 | 0 | 0 | 0 | 0 | 0 | 0 | 0 | 0 | 0 | 0 | 0 | 0 | 0 | 0 | 0 |
| $c^3$ | 0 | 0 | 0 | 1 | 0 | 0 | 1 | 0 | 0 | 0 | 0 | 0 | 0 | 0 | 0 | 0 | 0 | 0 | 0 | 0 | 0 | 0 | 0 | 0 | 0 | 0 | 0 | 0 | 0 | 0 | 0 |
| $c^4$ | 0 | 0 | 0 | 0 | 0 | 1 | 0 | 0 | 1 | 0 | 0 | 0 | 0 | 0 | 0 | 0 | 0 | 0 | 0 | 0 | 0 | 0 | 0 | 0 | 0 | 0 | 0 | 0 | 0 | 0 | 0 |
| $c^5$ | 0 | 0 | 0 | 0 | 0 | 0 | 0 | 1 | 0 | 0 | 0 | 1 | 0 | 0 | 0 | 0 | 0 | 0 | 0 | 0 | 0 | 0 | 0 | 0 | 0 | 0 | 0 | 0 | 0 | 0 | 0 |
| $c^6$ | 0 | 0 | 0 | 0 | 0 | 0 | 0 | 0 | 0 | 1 | 0 | 0 | 0 | 0 | 0 | 0 | 0 | 1 | 0 | 0 | 0 | 0 | 0 | 0 | 0 | 0 | 0 | 0 | 0 | 0 | 0 |
| $c^7$ | 0 | 0 | 0 | 0 | 0 | 0 | 0 | 0 | 0 | 0 | 1 | 0 | 0 | 0 | 0 | 0 | 0 | 0 | 0 | 0 | 1 | 0 | 0 | 0 | 0 | 0 | 0 | 1 | 0 | 0 | 0 |
| $c^8$ | 0 | 0 | 0 | 0 | 1 | 0 | 0 | 0 | 0 | 0 | 0 | 0 | 0 | 0 | 0 | 0 | 0 | 1 | 0 | 0 | 0 | 0 | 0 | 0 | 0 | 0 | 0 | 0 | 0 | 0 | 0 |
| $c^9$ | 0 | 0 | 0 | 0 | 0 | 1 | 0 | 0 | 0 | 0 | 0 | 0 | 0 | 0 | 0 | 1 | 0 | 0 | 0 | 0 | 0 | 0 | 0 | 0 | 1 | 0 | 0 | 0 | 0 | 0 | 0 |

Table 2: Associative Memory, Performance Simulation. Single pass convergence, based on the $T^{MEM}$ matrix given in Table 1. For each input, the Hamming distance to the closest stored memory vector is given.

**OOC Family: F = 31, K = 3**

Input Vector:

| $a$ | 1 | 1 | 0 | 1 | 0 | 0 | 0 | 1 | 0 | 0 | 0 | 0 | 0 | 0 | 1 | 0 | 0 | 0 | 0 | 1 | 0 | 0 |
|---|---|---|---|---|---|---|---|---|---|---|---|---|---|---|---|---|---|---|---|---|---|---|---|

Hamming distance from $c^1$ = 4

Output Vector:

| $c^1$ | 1 | 1 | 0 | 0 | 0 | 0 | 0 | 0 | 0 | 0 | 0 | 0 | 0 | 0 | 0 | 0 | 0 | 0 | 0 | 0 | 0 | 0 |
|---|---|---|---|---|---|---|---|---|---|---|---|---|---|---|---|---|---|---|---|---|---|---|---|

Input Vector:

| $b$ | 1 | 0 | 1 | 0 | 0 | 1 | 0 | 1 | 0 | 0 | 0 | 0 | 1 | 0 | 1 | 0 | 0 | 1 | 1 | 0 | 0 |
|---|---|---|---|---|---|---|---|---|---|---|---|---|---|---|---|---|---|---|---|---|---|---|

Hamming distance from $c^5$ = 0

Output Vector:

| $c^5$ | 0 | 0 | 0 | 0 | 0 | 0 | 1 | 0 | 0 | 0 | 0 | 1 | 0 | 0 | 0 | 0 | 0 | 0 | 0 | 0 | 0 |
|---|---|---|---|---|---|---|---|---|---|---|---|---|---|---|---|---|---|---|---|---|---|---|

Input Vector:

| $c$ | 1 | 0 | 0 | 1 | 0 | 0 | 1 | 0 | 0 | 1 | 0 | 1 | 0 | 0 | 0 | 1 | 0 | 0 | 1 | 0 |
|---|---|---|---|---|---|---|---|---|---|---|---|---|---|---|---|---|---|---|---|---|---|

Hamming distance from $c^4$ = 3
Hamming distance from $c^8$ = 3

Output Vector:

| $c^4 \otimes c^8$ | 0 | 0 | 0 | 1 | 0 | 0 | 1 | 0 | 0 | 0 | 0 | 1 | 0 | 0 | 0 | 0 | 0 | 0 | 1 | 0 |
|---|---|---|---|---|---|---|---|---|---|---|---|---|---|---|---|---|---|---|---|---|---|

Input Vector:

| $d$ | 0 | 1 | 0 | 0 | 0 | 0 | 0 | 0 | 0 | 0 | 0 | 0 | 0 | 0 | 0 | 0 | 0 | 0 | 0 | 0 | 0 | 0 |
|---|---|---|---|---|---|---|---|---|---|---|---|---|---|---|---|---|---|---|---|---|---|---|

Hamming distance from $c^1$ = 1

Output Vector:

| zero | 0 | 0 | 0 | 0 | 0 | 0 | 0 | 0 | 0 | 0 | 0 | 0 | 0 | 0 | 0 | 0 | 0 | 0 | 0 | 0 | 0 | 0 |
|---|---|---|---|---|---|---|---|---|---|---|---|---|---|---|---|---|---|---|---|---|---|---|

Table 4: Code Filtering (CDMA), Performance Simulation. Single pass convergence, based on the $T^{CDMA}$ matrix given in Table 3. Each input is specified by the intended message bit (1-bit = $c^k$; 0-bit = zero vector), plus the indicated interfering codes, arbitrarily shifted with respect to the message bit.

**OOC Family: F = 31, K = 3**

Input Vector = $[c^k \otimes c^1 \otimes c^2 \otimes c^4 \otimes c^5 \otimes c^7]$

| $a$ | 1 | 0 | 0 | 1 | 0 | 1 | 0 | 1 | 0 | 1 | 0 | 0 | 1 | 1 | 1 | 1 | 0 | 0 | 0 | 0 | 0 |
|---|---|---|---|---|---|---|---|---|---|---|---|---|---|---|---|---|---|---|---|---|---|---|

Output Vector: correct

| $c^k$ | 0 | 0 | 0 | 0 | 0 | 0 | 0 | 0 | 1 | 0 | 0 | 0 | 0 | 0 | 1 | 0 | 0 | 0 | 0 | 0 | 0 |
|---|---|---|---|---|---|---|---|---|---|---|---|---|---|---|---|---|---|---|---|---|---|---|

Input Vector = [zero $\otimes c^1 \otimes c^2 \otimes c^4 \otimes c^5 \otimes c^7$]

| $b$ | 1 | 0 | 0 | 0 | 0 | 1 | 0 | 1 | 0 | 0 | 0 | 1 | 1 | 1 | 0 | 0 | 1 | 1 | 0 | 0 |
|---|---|---|---|---|---|---|---|---|---|---|---|---|---|---|---|---|---|---|---|---|---|

Output Vector: correct

| [zero] | 0 | 0 | 0 | 0 | 0 | 0 | 0 | 0 | 0 | 0 | 0 | 0 | 0 | 0 | 0 | 0 | 0 | 0 | 0 | 0 | 0 |
|---|---|---|---|---|---|---|---|---|---|---|---|---|---|---|---|---|---|---|---|---|---|

Input Vector = [zero $\otimes c^1 \otimes c^2 \otimes c^3 \otimes c^4 \otimes c^5 \otimes c^7 \otimes c^8$]

| $c$ | 0 | 0 | 0 | 1 | 1 | 0 | 0 | 1 | 1 | 1 | 0 | 1 | 1 | 0 | 1 | 1 | 0 | 0 | 1 | 1 | 0 |
|---|---|---|---|---|---|---|---|---|---|---|---|---|---|---|---|---|---|---|---|---|---|

Output Vector: error

| $c^k$ | 0 | 0 | 0 | 0 | 0 | 0 | 0 | 1 | 0 | 0 | 0 | 0 | 0 | 1 | 0 | 0 | 0 | 0 | 0 | 0 | 0 |
|---|---|---|---|---|---|---|---|---|---|---|---|---|---|---|---|---|---|---|---|---|---|

| Statistical Behaviour | | | |
|---|---|---|---|
| | 1-bit | 0-bit | Total |
| Samples | 50037 | 50388 | 100425 |
| Errors | 0 | 5901 | 5901 |
| BER % | 0.00% | 11.71% | 5.88% |

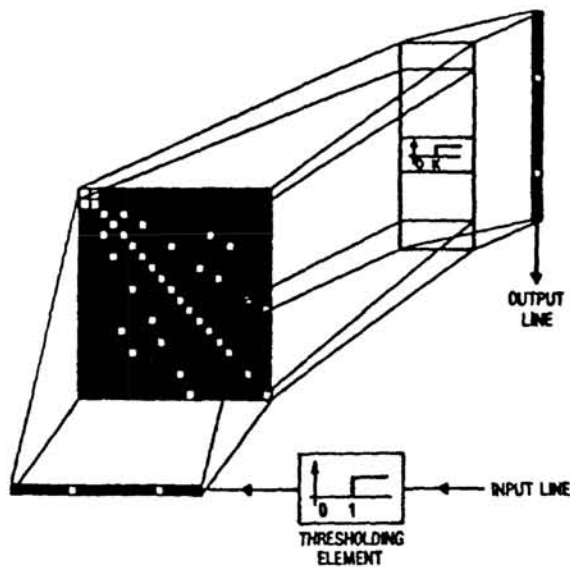

OUTPUT LINE

INPUT LINE

THRESHOLDING ELEMENT

Figure 1:
Schematic diagram of an optical neuromorphic processor using sparse Optical Orthogonal Codes. Notice the absence of feedback because of the single-pass convergence. The mask shown represents the realization of the content-addressable memory of Table 1.

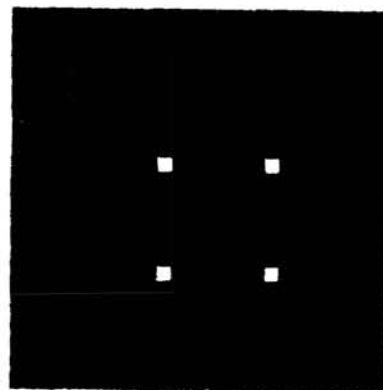

Figure 3:
Optical realization of a code filtering (CDMA) mask of Table 3. The 1's are represented by the transparent pixels, and the 0's by the opaque pixels.

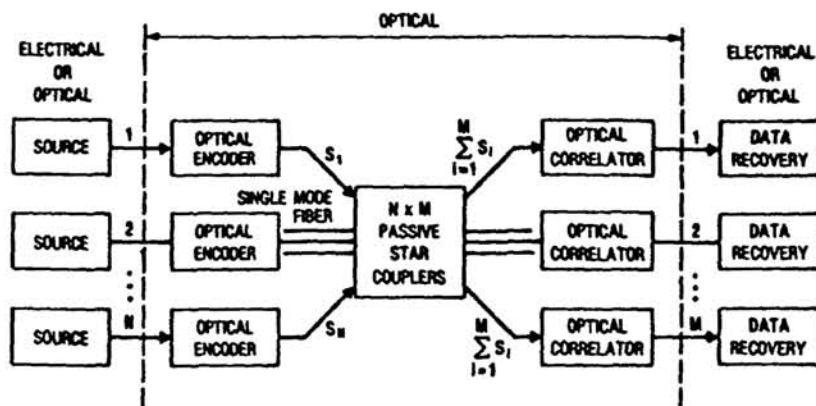

Figure 2:
Schematic diagram of a CDMA communications system over an Optical Fiber interconnection network. Each node represents one of the M possible distinct users in the system.